# Remote Sensing Image Analysis via a Texture Classification Neural Network

**Hayit K. Greenspan and Rodney Goodman**
Department of Electrical Engineering
California Institute of Technology, 116-81
Pasadena, CA 91125
hayit@electra.micro.caltech.edu

## Abstract

In this work we apply a texture classification network to remote sensing image analysis. The goal is to extract the characteristics of the area depicted in the input image, thus achieving a segmented map of the region. We have recently proposed a combined neural network and rule-based framework for texture recognition. The framework uses unsupervised and supervised learning, and provides probability estimates for the output classes. We describe the texture classification network and extend it to demonstrate its application to the Landsat and Aerial image analysis domain.

## 1  INTRODUCTION

In this work we apply a texture classification network to remote sensing image analysis. The goal is to segment the input image into homogeneous textured regions and identify each region as one of a prelearned library of textures, e.g. tree area and urban area distinction. Classification o f remote sensing imagery is of importance in many applications, such as navigation, surveillance and exploration. It has become a very complex task spanning a growing number of sensors and application domains. The applications include: landcover identification (with systems such as the AVIRIS and SPOT), atmospheric analysis via cloud-coverage mapping (using the AVHRR sensor), oceanographic exploration for sea/ice type classification (SAR input) and more.

Much attention has been given to the use of the spectral signature for the identifica-

tion of region types (Wharton, 1987; Lee and Philpot, 1991). Only recently has the idea of adding on spatial information been presented (Ton et al, 1991). In this work we investigate the possibility of gaining information from textural analysis. We have recently developed a texture recognition system (Greenspan et al, 1992) which achieves state-of-the-art results on natural textures. In this paper we apply the system to remote sensing imagery and check the system's robustness in this noisy environment. Texture can play a major role in segmenting the images into homogeneous areas and enhancing other sensors capabilities, such as multispectra analysis, by indicating areas of interest in which further analysis can be pursued. Fusion of the spatial information with the spectral signature will enhance the classification and the overall automated analysis capabilities.

Most of the work in the literature focuses on human expert-based rules with specific sensor data calibration. Some of the existing problems with this classic approach are the following (Ton et al, 1991):
- Experienced photointerpreters are required to spend a considerable amount of time generating rules.
- The rules need to be updated for different geographical regions.
- No spatial rules exist for the complex Landsat imagery.
An interesting question is if one can automate the rule generation. In this paper we present a learning framework in which spatial rules are learned by the system from a given database of examples.

The learning framework and its contribution in a texture-recognition system is the topic of section 2. Experimental results of the system's application to remote sensing imagery are presented in section 3.

## 2   The texture-classification network

We have previously presented a texture classification network which combines a neural network and rule-based framework (Greenspan et al, 1992) and enables both unsupervised and supervised learning. The system consists of three major stages, as shown in Fig. 1. The first stage performs feature extraction and transforms the image space into an array of 15-dimensional feature vectors, each vector corresponding to a local window in the original image. There is much evidence in animal visual systems supporting the use of multi-channel orientation selective band-pass filters in the feature-extraction phase. An open issue is the decision regarding the appropriate number of frequencies and orientations required for the representation of the input domain. We define an initial set of 15 filters and achieve a computationally efficient filtering scheme via the multi-resolution pyramidal approach.

The learning mechanism shown next derives a minimal subset of the above filters which conveys sufficient information about the visual input for its differentiation and labeling. In an unsupervised stage a machine-learning clustering algorithm is used to quantize the continuous input features. A supervised learning stage follows in which labeling of the input domain is achieved using a rule-based network. Here an information theoretic measure is utilized to find the most informative correlations between the attributes and the pattern class specification, while providing probability estimates for the output classes. Ultimately, a minimal representation for a library of patterns is learned in a training mode, following which the classification

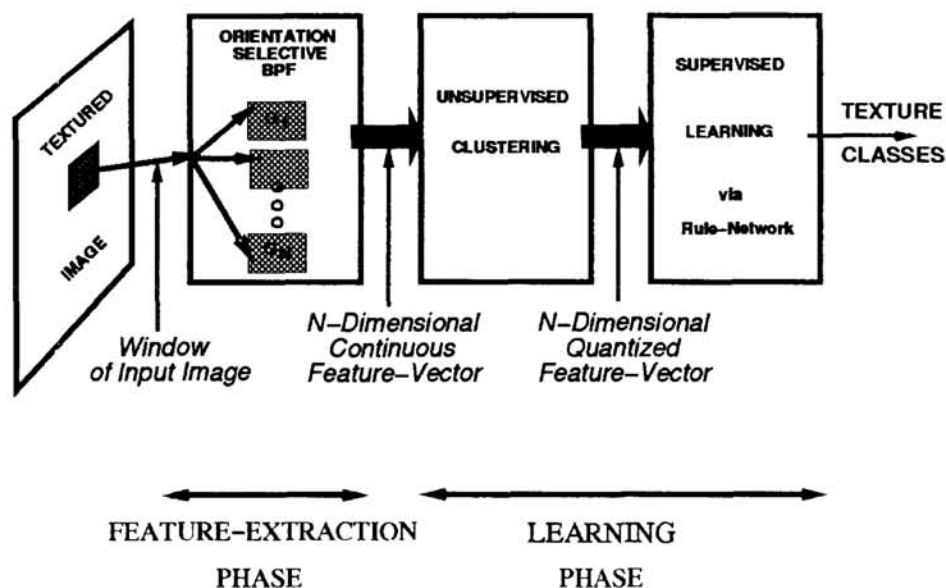

FEATURE-EXTRACTION        LEARNING

PHASE                        PHASE

Figure 1: System block diagram

of new patterns is achieved.

## 2.1  The system in more detail

The initial stage for a classification system is the feature extraction phase. In the texture-analysis task there is both biological and computational evidence supporting the use of Gabor-like filters for the feature-extraction. In this work, we use the Log Gabor pyramid, or the Gabor wavelet decomposition to define an initial finite set of filters. A computational efficient scheme involves using a pyramidal representation of the image which is convolved with fixed spatial support oriented Gabor filters (Greenspan at al, 1993). Three scales are used with 4 orientations per scale (0,90,45,135 degrees), together with a non-oriented component, to produce a 15-dimensional feature vector as the output of the feature extraction stage. Using the pyramid representation is computationally efficient as the image is subsampled in the filtering process. Two such size reduction stages take place in the three scale pyramid. The feature values thus generated correspond to the average power of the response, to specific orientation and frequency ranges, in an $8 * 8$ window of the input image. Each such window gets mapped to a 15-dimensional attribute vector as the output of the feature extraction stage.

The goal of the learning system is to use the feature representation described above to discriminate between the input patterns, or textures. Both unsupervised and supervised learning stages are utilized. A minimal set of features are extracted from the 15-dimensional attribute vector, which convey sufficient information about the visual input for its differentiation and labeling.

The unsupervised learning stage can be viewed as a preprocessing stage for achieving a more compact representation of the filtered input. The goal is to quantize the continuous valued features which are the result of the initial filtering, thus shifting to a more symbolic representation of the input domain. This clustering stage was found experimentally to be of importance as an initial learning phase in a classification system. The need for discretization becomes evident when trying to learn associations between attributes in a symbolic representation, such as rules.

The output of the filtering stage consists of $N$ ($=15$), continuous valued feature maps; each representing a filtered version of the original input. Thus, each local area of the input image is represented via an $N$-dimensional feature vector. An array of such $N$-dimensional vectors, viewed across the input image, is the input to the learning stage. We wish to detect characteristic behavior across the $N$-dimensional feature space, for the family of textures to be learned. In this work, each dimension, out of the 15-dimensional attribute vector, is individually clustered. All training samples are thus projected onto each axis of the space and one-dimensional clusters are found using the K-means clustering algorithm (Duda and Hart, 1973). This statistical clustering technique consists of an iterative procedure of finding K means in the training sample space, following which each new input sample is associated with the closest mean in Euclidean distance. The means, labeled 0 thru K minus 1 arbitrarily, correspond to discrete codewords. Each continuous-valued input sample gets mapped to the discrete codeword representing its associated mean. The output of this preprocessing stage is a 15-dimensional quantized vector of attributes which is the result of concatenating the discrete-valued codewords of the individual dimensions.

In the final, supervised stage, we utilize the existing information in the feature maps for higher level analysis, such as input labeling and classification. A rule - based information theoretic approach is used which is an extension of a first order Bayesian classifier, because of its ability to output probability estimates for the output classes (Goodman et al, 1992). The classifier defines correlations between input features and output classes as probabilistic rules. A data driven supervised learning approach utilizes an information theoretic measure to learn the most informative links or rules between features and class labels. The classifier then uses these links to provide an estimate of the probability of a given output class being true. When presented with a new input evidence vector, a set of rules R can be considered to "fire". The classifier estimates the posterior probability of each class given the rules that fire in the form $log(p(x)|R)$, and the largest estimate is chosen as the initial class label decision. The probability estimates for the output classes can now be used for feedback purposes and further higher level processing.

The rule-based classification system can be mapped into a 3 layer feed forward architecture as shown in Fig. 2 (Greenspan et al, 1993). The input layer contains a node for each attribute. The hidden layer contains a node for each rule and the output layer contains a node for each class. Each rule (second layer node $j$) is connected to a class via a multiplicative weight of evidence $W_j$.

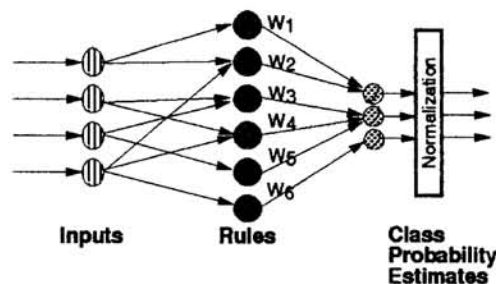

Figure 2: Rule-Based Network

## 3   Results

The above-described system has achieved state-of-the-art results on both structured and unstructured natural texture classification [5]. In this work we present initial results of applying the network to the noisy environment of satellite and air-borne imagery.

Fig. 3 presents two such examples. The first example (top) is an image of Pasadena, California, taken via the AVIRIS system (Airborne Visible/Infrared Imaging Spectrometer). The AVIRIS system covers 224 contiguous spectral bands simultaneously, at 20 meters per pixel resolution. The presented example is taken as an average of several bands in the visual range. In this input image we can see that a major distinguishing characteristic is urban area vs. hilly surround. These are the two categories we set forth to learn. The training consists of a 128*128 image sample for each category. The test input is a 512*512 image which is very noisy and because of its low resolution, very difficult to segment into the two categories, even to our own visual perception. In the presented output (top right), the urban area is labeled in white, the hillside in gray and unknown, undetermined areas are in darker gray. We see that a rough segmentation into the desired regions has been achieved. The probabilistic network's output allows for the identification of unknown or unspecified regions, in which more elaborate analysis can be pursued (Greenspan et al, 1992). The dark gray areas correspond to such regions; one example is the hill and urban contact (bottom right) in which some urban suburbs on the hill slopes form a mixture of the classes. Note that in the initial results presented the blockiness perceived is the result of the analysis resolution chosen. Fusing into the system additional spectral bands as our input, would enable pixel resolution as well as enable detecting additional classes (not visually detectable), such as concrete material, a variety of vegetation etc.

A higher resolution Airborne image is presented at the bottom of Fig. 3. The classes learned are bush (output label dark gray), ground (output label gray) and a structured area, such as a field present or the man-made structures (white). Here, the training was done on 128*128 image examples (1 example per class). The input image is 800*800. In the result presented (right) we see that the three classes have been found and a rough segmentation into the three regions is achieved. Note in particular the detection of the bush areas and the three main structured areas in the image, including the man-made field, indicated in white.

Our final example relates to an autonomous navigation scenario. Autonomous vehicles require an automated scene analysis system to avoid obstacles and navigate through rough terrain. Fusion of several visual modalities, such as intensity-based segmentation, texture, stereo, and color, together with other domain inputs, such as soil spectral decomposition analysis, will be required for this challenging task. In Fig. 4. we present preliminary results on outdoor photographed scenes taken by an autonomous vehicle at JPL (Jet Propulsion Laboratory, Pasadena). The presented scenes (left) are segmented into bush and gravel regions (right). The training set consists of 4 64 * 64 image samples from each category. In the top example (a 256*256 pixel image), light gray indicates gravel while black represents bushy regions. We can see that intensity alone can not suffice in this task (for example, top right corner). The system has learned some textural characteristics which guided

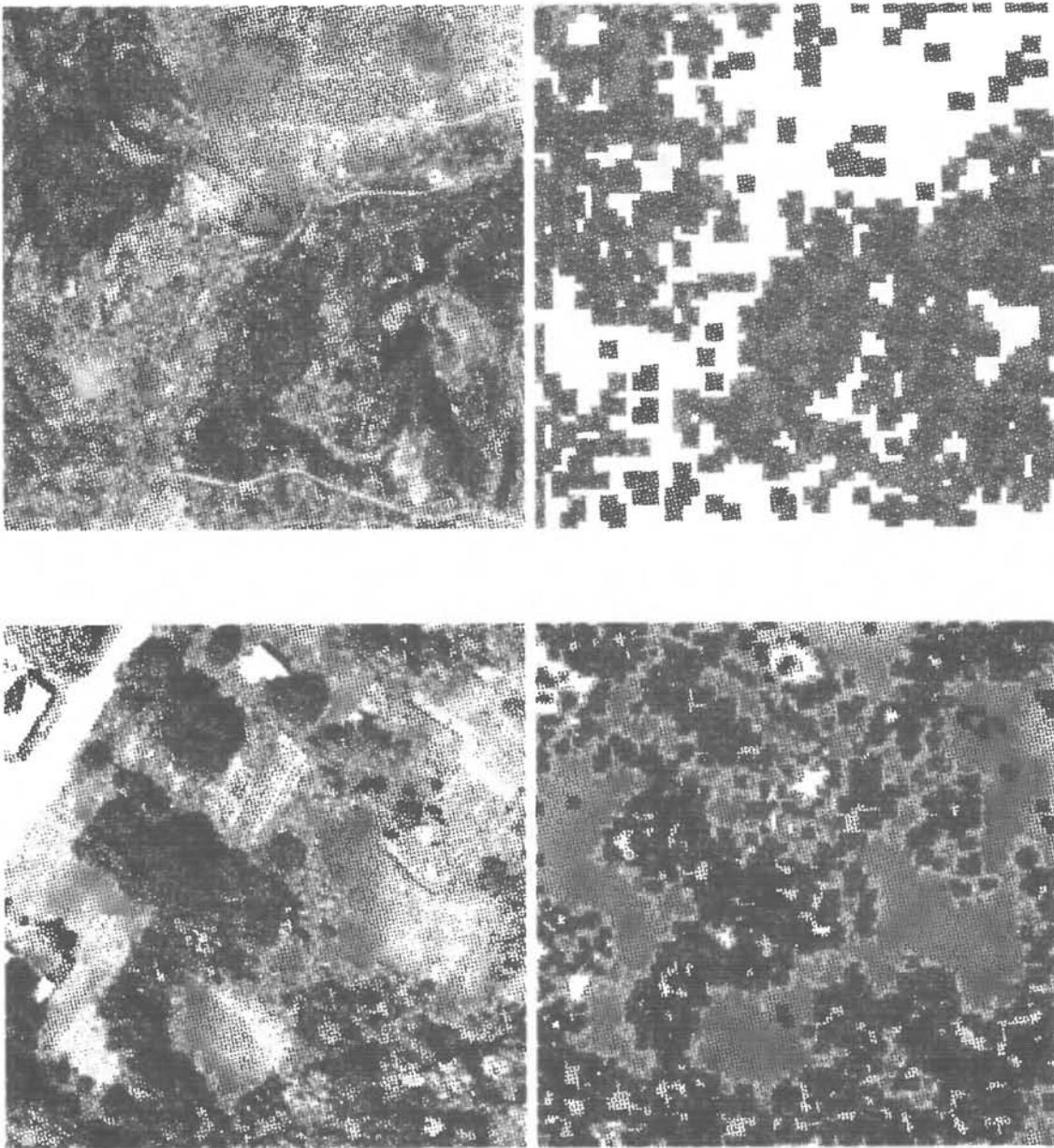

Figure 3: Remote sensing image analysis results. The input test image is shown (left) followed by the system output classification map (right). In the AVIRIS (top) input, white indicates urban regions, gray is a hilly area and dark gray reflects undetermined or different region types. In the Airborne output (bottom), dark gray indicates a bush area, light gray is a ground cover region and white indicates man-made structures. Both robustness to noise and generalization are demonstrated in these two challenging real-world problems.

the segmentation in otherwise similar-intensity regions. Note that this is also probably the cause for identifying the track-like region (e.g., center bottom) as bush regions. We could learn track-like regions as a third category, or specifically include such examples as gravel in our training set.

In the second example (a 400*400 input image, bottom) light gray indicates gravel, dark gray represents a bush-like region, and black represents the unknown category. Here, the top right region of the sky, is labeled correctly as an unknown, or new category. Note that intensity alone would have confused that region as being gravel. Overall, the texture classification neural-network succeeds in achieving a correct, yet rough, segmentation of the scene based on textural characteristics alone. These are encouraging results indicating that the learning system has learned informative characteristics of the domain.

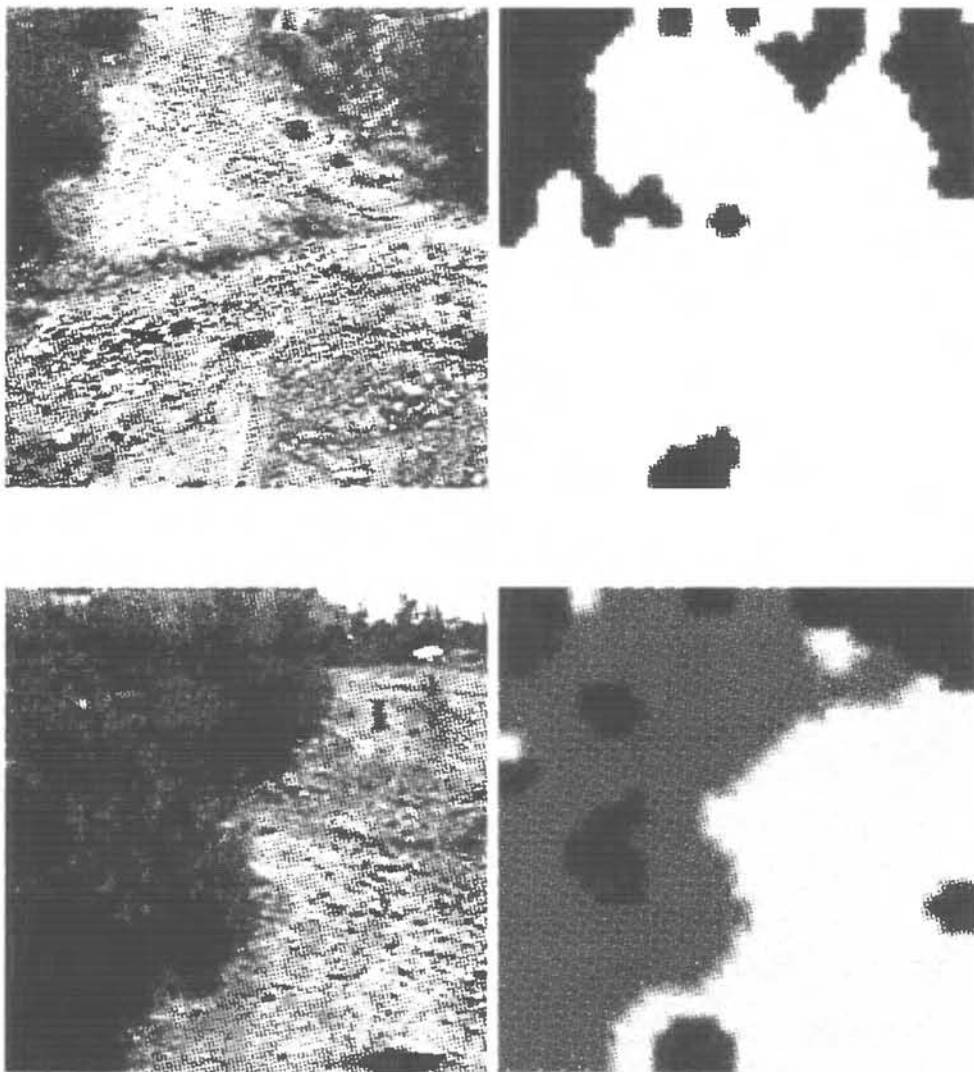

Fig 4: Image Analysis for Autonomous Navigation

## 4    Summary and Discussion

The presented results demonstrate the network's capability for generalization and robustness to noise in very challenging real-world problems. In the presented framework a learning mechanism automates the rule generation. This framework can answer some of the current difficulties in using the human expert's knowledge. Further more, the automation of the rule generation can enhance the expert's knowledge regarding the task at hand. We have demonstrated that the use of textural spatial information can segment complex scenery into homogeneous regions. Some of the system's strengths include generalization to new scenes, invariance to intensity, and the ability to enlarge the feature vector representation to include additional inputs (such as additional spectral bands) and learn rules characterizing the integrated modalities. Future work includes fusing several modalities within the learning framework for enhanced performance and testing the performance on a large database.

### Acknowledgements

This work is supported in part by Pacific Bell, and in part by DARPA and ONR under grant no. N00014-92-J-1860. H. Greenspan is supported in part by an Intel fellowship. The research described in this paper was carried out in part by the Jet Propulsion Laboratories, California Institute of Technology. We would like to thank Dr. C. Anderson for his pyramid software support and Dr. L. Matthies for the autonomous vehicle images.

### References

S. Wharton. (1987) A Spectral-Knowledge-Based Approach for Urban Land-Cover Discrimination. *IEEE Transactions on Geoscience and Remote Sensing*, Vol. GE-25[3]:272-282.

J. Lee and W. Philpot. (1991) Spectral Texture Pattern Matching: A Classifier For Digital Imagery. *IEEE Transactions on Geoscience and Remote Sensing*, Vol. 29[4]:545-554.

J. Ton, J. Sticklen and A. Jain. (1991) Knowledge-Based Segmentation of Landsat Images. *IEEE Transactions on Geoscience and Remote Sensing*, Vol. 29[2]:222-232.

H. Greenspan, R. Goodman and R. Chellappa. (1992) Combined Neural Network and Rule-Based Framework for Probabilistic Pattern Recognition and Discovery. In J. E. Moody, S. J. Hanson, and R. P. Lippman (eds.), *Advances in Neural Information Processing Systems 4.*, 444-452, San Mateo, CA: Morgan Kaufmann Publishers.

H. Greenspan, R. Goodman, R. Chellappa and C. Anderson. (1993) Learning Texture Discrimination Rules in a Multiresolution System. Submitted to *IEEE Transactions on Pattern Analysis and Machine Intelligence.*

R. O. Duda and P. E. Hart. (1973) *Pattern Classification and Scene Analysis.* John Wiley and Sons, Inc.

R. Goodman, C. Higgins, J. Miller and P. Smyth. (1992) Rule-Based Networks for Classification and Probability Estimation. *Neural Computation*, [4]:781-804.